# DeltaDEQ: Exploiting Heterogeneous Convergence for Accelerating Deep Equilibrium Iterations

**Zuowen Wang  Longbiao Cheng  Pehuen Moure  Niklas Hahn  Shih-Chii Liu**
Institute of Neuroinformatics, University of Zurich and ETH Zurich
`{zuowen, shih}@ini.uzh.ch`

## Abstract

Implicit neural networks including deep equilibrium models have achieved superior task performance with better parameter efficiency in various applications. However, it is often at the expense of higher computation costs during inference. In this work, we identify a phenomenon named **heterogeneous convergence** that exists in deep equilibrium models and other iterative methods. We observe much faster convergence of state activations in certain dimensions therefore indicating the dimensionality of the underlying dynamics of the forward pass is much lower than the defined dimension of the states. We thereby propose to exploit heterogeneous convergence by storing past linear operation results (e.g., fully connected and convolutional layers) and only propagating the state activation when its change exceeds a threshold. Thus, for the already converged dimensions, the computations can be skipped. We verified our findings and reached 84% FLOPs reduction on the implicit neural representation task, 73% on the Sintel and 76% on the KITTI datasets for the optical flow estimation task while keeping comparable task accuracy with the models that perform the full update.

## 1   Introduction

Implicit neural networks [2, 13] have gained much attention in recent years. They are capable of matching or surpassing state-of-the-art performance in domains such as computer vision [3, 4, 17, 32, 41, 46], language modeling [2] and audio processing [32, 35]. Implicit networks typically model these tasks with certain dynamics represented by the evolution of the intermediate hidden states and are often described as fixed-point equations or differential equations. The implicit models, represented by the deep equilibrium (DEQ) model family, often achieve better parameter efficiency for reaching a similar level of task accuracy as their regular neural network counterparts. Furthermore, the implicit function theorem assists implicit models in avoiding the need for direct differentiation through numerous forward steps, like those required in back-propagation through time, which demands substantial memory for training.

Despite their compact architectural design with fewer parameters, implicit models frequently require extensive computations during inference to better approximate the fixed point. The fixed point is typically computed with fixed-point iteration methods [4] or other root-solving techniques such as Broyden [7] or Anderson [54] methods, all of which require multiple forward passes of the model. The substantial computational cost at inference time significantly hinders the deployment of implicit models. Various strategies have been developed to reduce this computational burden, including fixed-point reuse [4, 41] to accelerate the convergence of the forward pass for temporally correlated inputs like consecutive video frames, and global early stopping criteria [4] that halt the forward pass once the absolute or relative change of the hidden states norm is below a threshold. However, these approaches uniformly control the termination of updates across all dimensions of the hidden states and they fail to consider the internal structure of the dynamics.

This work makes a finer-grained inspection of the dynamics of the hidden states update. By analyzing the element-wise trajectory and dimensionality of hidden states update, we observed the **heterogeneous convergence** phenomenon among implicit models. That is, there exists a significant variance in convergence speeds across different dimensions of the hidden states; and that certain dimensions converge substantially faster than others in the forward pass of an implicit model. Based on this finding, we modified the forward pass update of the DEQ models with the delta updating rule that stores the intermediate results of computationally intensive linear operators and only calculates at dimensions where their changes between two sequential updates exceed a threshold. Our method is orthogonal to other acceleration techniques including fixed-point reuse [4, 32] and early stopping [4, 41] of iterations. Unlike fixed-point reuse acceleration, our method does not assume temporal correlation of inputs i.e. effective even for static. While primarily applied to DEQ-based methods, our technique can be adapted to any iterative method that uses a fixed-point search in the forward pass. We empirically tested our method for two tasks: implicit neural representation (INR) for images and optical flow (OF) estimation. Compared to previous DEQ-based methods, Our approach achieved an 84% reduction in FLOPs for the INR task and a 73% and 76% reduction for the OF task using the Sintel and KITTI datasets, respectively, without significant loss in task accuracy. The code is available at https://github.com/ZuowenWang0000/Delta-Deep-Equilibrium-Models.

## 2   Background

The majority of the findings in this study are derived from deep equilibrium models (DEQ)[1], which represent a specific category of implicit models. We offer essential annotations and background information on DEQ in this section. A DEQ layer is modeled as a fixed-point equation:

$$z^* = f_\theta(z^*, x), \tag{1}$$

where $x$ is the input and the layer is parameterized by $\theta$. Notice that $f_\theta$ does not need to be single-layered but could also be a block of layers. Thus, $z^*$ could be activations of a block of layers. This fixed-point equation is solved with Broyden method [7] or Anderson acceleration [54] by finding the root of $f_\theta(z^*, x) - z^* = 0$ in the initial DEQ works [2, 4].

Alternatively, the fixed-point equation can be solved by iterative methods. Picard's method is an iterative method for approximating the fixed point $z^*$. By simply substituting the state $z^i$ at $i$-th iteration as the input and $z^0$ as initialization, the state at iteration step $i + 1$ is computed as

$$z^{i+1} = f_\theta(z^i, x), \tag{2}$$

and $z^i \to z^*$ as $i \to \infty$ when $f_\theta$ is a contraction mapping. Empirically the $z^*$ can be approximated in limited numbers of iterations, in other words, the fixed-point iteration (FPI) converges empirically in limited steps.

For a temporal input sequence $x_t$ where $t = 1, ..., T$, processing each input $x_t$ at time $t$ requires solving a corresponding fixed-point $z_t^* = f_\theta(z_t^*, x_t)$. This means that for every input timestep $t$, the forward pass of DEQ involves solving a fixed-point equation which could be computationally intensive. From previous studies [4, 41, 56], reusing the fixed point $z_{t-1}^*$ as the initialization ($z_t^0 = z_{t-1}^*$) for processing the next timestep input $x_t$, could accelerate the convergence speed to $z_t^*$.

## 3   Observations and motivation

Our methods are based on several key observations on the forward pass of DEQ models. We first demonstrate the observations with a simple DEQ model which is formulated as a deep equilibrium layer $z^* = \tanh(W_z \cdot z^* + W_x \cdot x + b)$ and a linear readout layer $y = Wz^*$, where $W_z \in \mathbb{R}^{20 \times 20}, W_x \in \mathbb{R}^{20 \times 1}, W \in \mathbb{R}^{1 \times 20}$, and $b \in \mathbb{R}^{20 \times 1}$ are the weights and bias respectively. We generated a dataset that consists of data points $(x, \sin(x))$ with $x$ being evenly sampled 200 points in $[-2\pi, 2\pi]$. We train the network to fit the $\sin(x)$ function learning from these data samples. The forward pass is implemented with Picard's method in Eq. 2 with 15 iterations and the backward pass is implemented with back-propagation through time (BPTT). The loss function used for training is

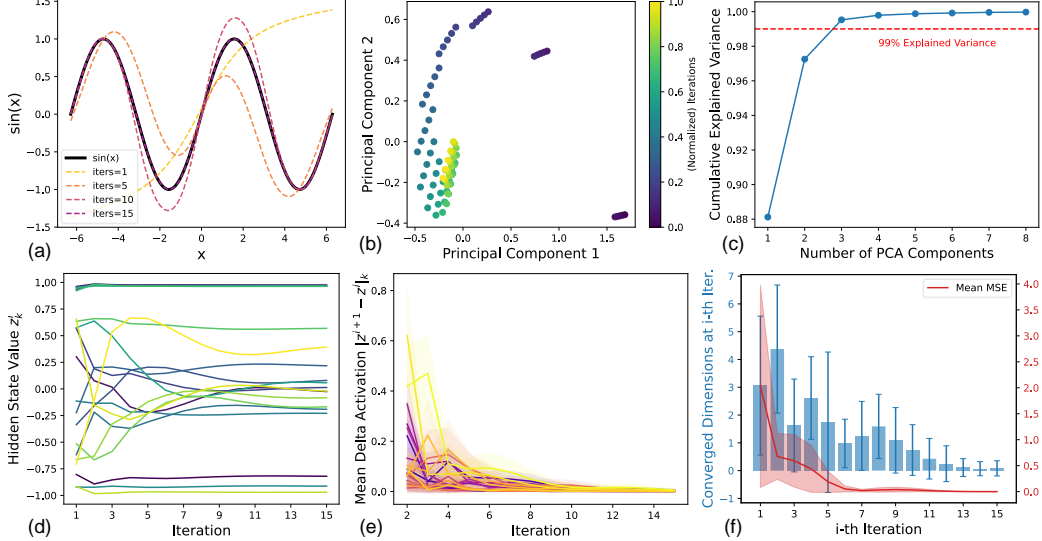

Figure 1: (a) Reconstruction evolution when increasing inference iterations. (b) Hidden states trajectory for 5 consecutive input points with the first two principal components. Details in A.1. (c) Cumulative explained variance for all hidden states. (d) Evolution of different dimensions of hidden states (represented by colors) over iterations. (e) Mean delta activation for different dimensions (represented by colors). The colored solid areas indicate the standard deviation from different inputs. (f) Histogram of converged dimensions (blue) at i-th iteration and evolution of the model prediction MSE (red).

$\ell = \ell_{\mathrm{MSE}} + \ell_{||\cdot||_2}$ where $\ell_{\mathrm{MSE}}$ is the mean squared error. The second term $\ell_{||\cdot||_2} = \lambda \cdot (||W_z||_2 - 1)$ is the spectral norm regularization on the matrix $W_z$ and is applied when $||W_z||_2 > 1$ in each iteration. This regularization is to ensure the model realizes a contrastive mapping and guarantees stable fixed points as stated in the Banach Fixed Point Theorem [6]. As shown in Fig. 1(a) the model gradually converges to the ground truth $\sin(x)$ values when the iterations at inference increased from 1 to 15.

In Fig. 1(c), we concatenated hidden states from different iterations for all the input $x$ in the dataset and conducted the principal component analysis (PCA) on them. We can see that although the hidden states are defined with 20 dimensions, with only 3 principal components we could explain over 99% of the variance in all hidden states trajectory. This observation shows that for DEQ models trained on specific tasks, the underlying dynamics could be depicted with much lower dimensionalities than the defined state $z^*$, which was observed similarly in [50] for continuous RNNs. Details of the PCA method are in Appendix. A.1.

**Heterogeneous convergence** The underlying dynamics of DEQ models, as well as other forms of recurring architectures [37], often have lower intrinsic dimensions than their explicitly defined dimensionality. Fig. 1(d) shows how this low intrinsic dimension characteristic is reflected in the original hidden state space. We can see that some dimensions of $z$ experience much larger fluctuation with the hidden state trajectory while certain dimensions converge within a few iterations. Fig. 1(e) presents the evolution of the mean delta activation $|z^{i+1} - z^i|_k$ at dimension $k$ for 1000 uniformly sampled input points. We can observe very different convergence speed for different dimensions and large variances in all dimensions. The averaged histogram statistics for the number of dimensions that converged at $i - th$ iteration is shown in Fig. 1(f). We observe that although globally the model reaches an MSE plateau at around 8 iterations, many dimensions have already converged long before that. We name this phenomenon **heterogeneous convergence**, indicating that different dimensions of the state $z$ converge at uneven speeds for a fixed-point iteration implemented forward pass of a DEQ. A natural question arises from observations of this phenomenon:

*Can we exploit the heterogeneous convergence phenomenon in the dynamics of hidden state updates for accelerating deep equilibrium models and other iterative methods?*

Indeed, identifying and leveraging the rates at which dimensions stabilize could lead to more efficient computational strategies. In Sec. 4 we describe our DeltaDEQ method that exploits the heterogeneous convergence for reducing computation of the DEQ forward pass together with details for other associated methods used in this work.

# 4   Methods

We propose DeltaDEQ to leverage the heterogeneous convergence phenomenon by inducing the **delta activation sparsity** in fixed-point iterations. DeltaDEQ can be instantiated with mainstream network architectures including recurrent neural networks (RNNs), convolutional neural networks (CNNs) and transformers just like DEQ. Computation savings for DeltaDEQ do **not** require the strong assumption [44, 45] that input data has to be minimally changing temporal sequences. Even when processing a static input, DeltaDEQ can still save computation in comparison to DEQ due to the convergence nature of equilibrium evolution.

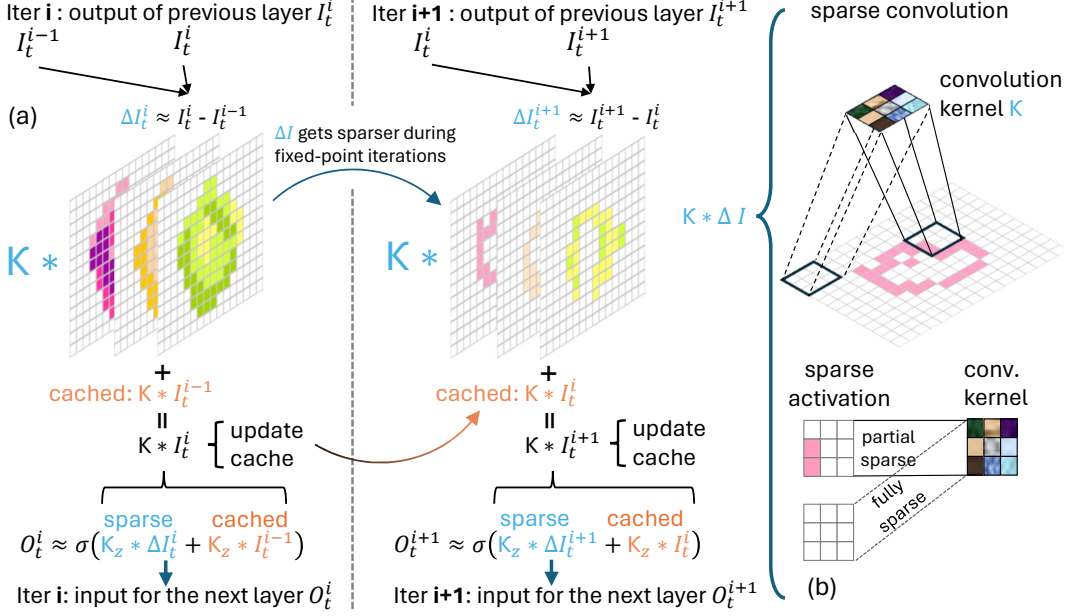

Figure 2: (a) Convolution type of DeltaDEQ. The input $I_t^{i-1}$ from the previous iteration is stored and subtracted to create the sparse $\Delta I_t^i$. White represents zero. (b) For sparse convolution, in theory, all zero entries in the feature map can be skipped; in practice, this is more feasible on hardware [1, 15] when the entire activation patch is fully sparse. The complete formulation and pseudo-code are given in A.2 and RNN type DeltaDEQ is illustrated in A.5.1.

## 4.1   Delta deep equilibrium layer (DeltaDEQ)

We start the introduction of the DeltaDEQ with a simple instantiation of a single RNN layer:
$$z_t^{i+1} = \sigma(W_z \cdot z_t^i + W_x \cdot x_t), \tag{3}$$
where $x_t \in \mathbb{R}^{d_x}$ is the input at timestep $t$, $z_t^i \in \mathbb{R}^d$ is the hidden states vector at fixed-point iteration $i$, $W_z \in \mathbb{R}^{d \times d}$ and $W_x \in \mathbb{R}^{d \times d_x}$ are the weight matrices and $\sigma$ represents the activation function. Due to the linearity of $W_z$ projection, Eq. 3 can be reformulated as follows:
$$z_t^{i+1} = \sigma(W_z \cdot (z_t^i - z_t^{i-1}) + W_z \cdot z_t^{i-1} + W_x \cdot x_t). \tag{4}$$
Assuming that $z_t^i - z_t^{i-1}$ is sparse, namely many dimensions are zeros, then the computation of the matrix-vector multiplication can be greatly reduced (see Fig. 5). This sparsity assumption holds due to the fact that DEQ models are trained to converge in the forward pass. In order to achieve higher sparsity, we further apply an **element-wise delta threshold** $\tau$ to zero out small changes. We define the thresholded delta hidden states vector and apply the delta rule on Eq. 4 as follows:

$$\Delta z_t^i := \begin{cases} z_t^i - z_t^{i-1} & \text{if } |z_t^i - z_t^{i-1}| \geq \tau \\ 0 & \text{otherwise} \end{cases} \quad \textbf{(Delta Rule)} \tag{5}$$

$$z_t^{i+1} \stackrel{(5)}{\approx} \sigma(\underbrace{W_z \cdot \Delta z_t^i}_{\text{sparse}} + \underbrace{W_z \cdot z_t^{i-1}}_{\text{cached in } C_z} + \underbrace{W_x \cdot x_t}_{\text{cached in } C_x}). \tag{6}$$

$$C_z \leftarrow W_z \cdot \Delta z_t^i + C_z \quad \text{(update cache)} \tag{7}$$

Notice that Eq. 5 operates element-wise. The assignment Eq. 7 states that after finishing the sparse matrix-vector multiplication, we update the stored value $C_z$ for the next iteration. There is no need to update the cached value $C_x$ within the same input timestep $t$ since the input $x_t$ does not change. But when the computation moves to $x_{t+1}$, a similar update rule as in Eq. 5 can be applied on input and thus also saving computation. An illustration for the RNN type DeltaDEQ is in Sec. A.5.1 Fig. 5.

**Convolutional DeltaDEQ** The convolution layers, which are most commonly used for vision tasks, are in principle also composed of linear transformation followed by non-linearities. Thus, we can also reformulate a convolutional DEQ block as a convolutional DeltaDEQ (ConvDeltaDEQ) block. Assuming the input features to a layer in a DeltaDEQ block at iteration $i$ is $I_t^i$ and the output is $O_t^i$:

$$O_t^i = \sigma(\mathcal{K} * I_t^i) = \sigma(\mathcal{K} * (I_t^i - I_t^{i-1} + I_t^{i-1})) \overset{(5)}{\approx} \sigma(\underbrace{\mathcal{K} * \Delta I_t^i}_{\text{sparse}} + \underbrace{\mathcal{K} * I_t^{i-1}}_{\text{cached}}), \tag{8}$$

where $*$ denotes the convolution operation and $\mathcal{K}$ is the convolution kernel. The convolution $\mathcal{K} * \Delta I_t^i$ is conducted on the sparse delta feature map $\Delta I_t^i$. The multiply-accumulate (MAC) operations could be skipped on the zero entries in $\Delta I_t^i$ as illustrated in Fig. 2.

**Theoretical complexity analysis and hardware practicality** Assuming the sparsity level (% zero entries) in $z_t^i$ is $o_{sp}$. For every fixed-point iteration for the RNN type of network layer as in Eq. 5 the floating point operations (FLOPs) spent on computing $W_z \cdot z_t^i$ is $2d^2$, while the delta thresholded version costs approx. $2(1 - o_{sp})d^2 + d$. For convolution kernel with kernel size $k$ and stride $s$, and assuming $\Delta z_t^i, \ z_t^i \in \mathbb{R}^{d \times d}$ and padding so that the output feature map still has size $d \times d$, the original convolution costs approx. $2d^2 k^2$ FLOPs while the delta version costs approx. $2(1 - o_{sp})d^2 k^2 + d^2$ FLOPs. A large level of $o_{sp}$ could reduce FLOPs greatly for both the RNN type of layer as well as a convolution in theory. However, while the computation cost reduction for the RNN type of layer is easy to realize in practice [20, 43], the sparse convolution is in general more difficult due to its fine granularity and its realization often requires special hardware or library support [1, 15].

## 4.2 Fixed-point iteration instead of root solving

The original DEQ work adopted root-solving, including Broyden [7] and Anderson [54] methods, in the forward pass. These methods contain overhead computation other than the model inference $f_\theta(z_t^i, x_t)$ itself. We found that fixed-point iterations, which include Picard's iteration (Eq. 2) and Krasnoselskii–Mann (KM) iteration, suffice for approximating the fixed-point in the forward pass and has good (global) convergence speed (fewer iterations needed to converge) in comparison to root-solving techniques. KM iteration maintains a history of one past iteration and conducts weighted sum: $z_t^{i+1} = \alpha_t f_\theta(z_t^i, x_t) + (1 - \alpha_t)z_t^i$, where $\alpha_t$ is the coefficient for the weighted sum and in our work we choose a fixed $\alpha$ namely $\alpha_t = \alpha$. The KM method helps to accelerate the overall convergence of the fixed-point iteration with better asymptotic and stabilize the trajectory of the forward pass and improve task performance. It does not induce an additional memory footprint with DeltaDEQ since the storage of two steps of states is already required. We provide a detailed comparison of fixed-point iterations and root-solving techniques in A.7.

## 4.3 Other acceleration techniques for DeltaDEQ

We adopt another two major acceleration techniques from previous works [2, 4, 32, 41, 56] for DeltaDEQ: (1) fixed-point reuse and (2) global early stopping. The fixed-point reuse method is often used for processing temporal sequences such as video frames. It initializes the hidden states $z_{t+1}^0$ with the fixed-point from last input timestep $t$, namely $z_{t+1}^0 = z_t^*$. This technique is based on the assumption that small distances between consecutive inputs $d(x_t, x_{t+1})$ will induce small distances of fixed-point $d(z_t^*, z_{t+1}^*) \ll d(z_{t+1}^0, z_{t+1}^*)$ under their corresponding metric spaces. In the work [32] the authors recycle the fixed point from the last training epoch as initialization for accelerating the training. Distinct from heterogeneous convergence, global early stopping is used to terminate the fixed-point iteration when it has converged **globally**. After computing the new hidden state $z_t^i$, we calculate its absolute $\|z_t^i - z_t^{i-1}\|_2$ or relative $\|z_t^i - z_t^{i-1}\|_2 / \|z_t^{i-1}\|_2$ distance to the previous hidden state $z_t^{i-1}$. If the distance is smaller than the tolerance, the forward pass is terminated. We emphasize that global early stopping requires storing the previous hidden state for calculating the difference $z_t^i - z_t^{i-1}$, which is also required for the delta rule. This additional computation and memory usage can be shared for both techniques.

### 4.4 Training and fine-tuning DeltaDEQ

One of the biggest advantages of deep equilibrium models [2] is the constant memory and computation costs for training in comparison to explicit iterative methods with (truncated-)BPTT. The loss value $\ell$ evaluated at fixed-point $z^*$ in Eq. 1 with respect to the function parameters $\theta$ is given by the implicit function theorem (IFT) [2, 36]:

$$\frac{\partial \ell(y_{\text{gt}}, r_\beta(z^*))}{\partial \theta} = \frac{\partial \ell(y_{\text{gt}}, r_\beta(z^*))}{\partial z^*}\frac{\partial z^*}{\partial \theta} = \frac{\partial \ell(y_{\text{gt}}, r_\beta(z^*))}{\partial z^*}\left(I - \frac{\partial f_\theta(z^*, x)}{\partial z^*}\right)^{-1}\frac{\partial f_\theta(z^*, x)}{\partial \theta}, \quad (9)$$

where $r_\beta$ is the final read-off layer with parameters $\beta$ and $y_{\text{gt}}$ is the ground truth label. The inversion in Eq. 9 could be expensive when the dimensionality of $z^*$ is large. However, various methods have been proposed to avoid computing the inversion [19, 26, 28] without hurting task performance much. The Eq. 9 shows that the backward pass is independent of the forward pass regardless. Thus, applying the delta rule (Eq. 5) in the forward pass **will not** impose any additional cost in the backward pass.

## 5 Experiments

In this section, we present the experimental results of the INR task and the OF task with the DeltaDEQ method. We showcase the two instantiations: RNN and CNN-based DeltaDEQs and present the computation reduction we could achieve without hurting the task accuracy.

### 5.1 Implicit neural representation

In this section, we present empirical results of DeltaDEQ for implicit neural representation (INR) [32, 48, 51]. INR is a type of method that represents the data with a coordination-based neural network. For example, for encoding a 2D image, the network is trained with input coordinate $(x, y)$ and to predict the corresponding (R,G,B) or greyscale value of the pixel $(x, y)$. The pixels in an image are the training set for fitting the network to represent such high-frequency data. Despite the simplicity of the task setting, training INR models to reconstruct the original data with good quality is not easy.

Two of the representative works are Siren [48] and multiplicative filter networks with Fourier filters [51]. A later work [32] modified these two methods with a deep equilibrium paradigm and achieved better task performance with the same amount of parameters. The experimental setup of DeltaDEQ for INR is largely based on [32]. Methods and hardware details and additional results are in A.5.

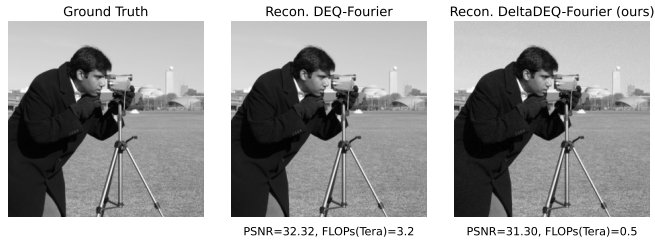

Ground Truth     Recon. DEQ-Fourier     Recon. DeltaDEQ-Fourier (ours)

PSNR=32.32, FLOPs(Tera)=3.2     PSNR=31.30, FLOPs(Tera)=0.5

Figure 3: Original image and reconstructions with INR network.

**DeltaDEQ architectures for INR** We apply the delta rule on the Implicit Sine-activate Networks and Implicit Multiplicative Filter Networks with Fourier filters, which we name DEQ-Siren and DEQ-Fourier respectively and their delta version DeltaDEQ-Siren and DeltaDEQ-Fourier. The DEQ-Siren (Eq. 10) and DEQ-Fourier (Eq. 11) are formulated as follows:

$$z^\star = \text{Sin}\left(Wz^\star + W\text{Sin}(Vx) + Ux + b\right) \qquad (10)$$

$$z^\star = (Wz^\star + Wg(x; V) + b) \circ g(x; U) \qquad (11)$$

where $g(x; U) := \text{Sin}(Ux)$ represents the Fourier filters [51]. The dimensionalities of the hidden state $z$ used in the experiments is 512. The most computationally heavy part is the dense matrix-vector multiplication $Wz^*$. We apply the delta rule described in Eq. 5 to convert the forward pass to reduce FLOPs (detailed formulations specific for each architecture are given in A.5.3).

**Inference acceleration with DeltaDEQ** We first train vanilla DEQ-Siren and DEQ-Fourier models without incorporating the delta rule in the training stage. The hidden state size is 512 and spectral normalization [42] is used for better stability and to ensure the non-diverging behavior of the network. The input image size is $512 \times 512$. As in [32], we use phantom gradient [26] to accelerate DEQ training. We use Adam [34] optimizer and cosine annealing learning rate schedule [38] with an initial

learning rate of 0.001. The fixed point from the end of one training epoch is used for the initialization of the next epoch, and only one forward iteration is used if the fixed point is reused. This training technique [32] greatly accelerates the training speed and saves a lot of FLOPs. For global early stopping, we use absolute distance with a tolerance of 0.001 and a maximum of 40 forward iterations.

Figure 4 demonstrate the relationship of computation saving (FLOPs Reduction (%) ↑), reconstruction quality (peak signal-to-noise ratio, PSNR ↑) w.r.t. the choice of inference delta threshold as in Eq. 5. Mean values of three runs are given and the standard deviations are shown as colored areas around the mean. We evenly sample 1000 inference delta threshold values between 0 and 0.1, and 100 threshold values between 0.1 to 0.5. Both DeltaDEQ instantiations show great computation savings (55% to 80% approx.) with a robust range of delta threshold levels ($10^{-3}$ to 0.5) with little change in the task performance accuracy. Figure 3 shows a qualitative comparison of the reconstructed image with DEQ-Fourier and our

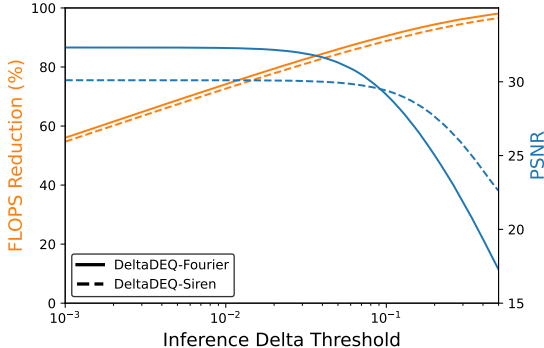

Figure 4: FLOPs reduction and task accuracy (PSNR) at different inference delta threshold.

DeltaDEQ-Fourier with threshold 0.5. We can see DeltaDEQ-Fourier achieved $\approx 84\%$ FLOPs reduction with a minor drop in PSNR.

**Learning with delta for the forward pass of DEQ** The forward pass of training DeltaDEQ is decoupled from the gradient calculation as described in Sec. 4.4. Thus, the saving of the forward pass could directly contribute to the training time reduction of DEQ-based INR methods. We study in this section the model behaviors when incorporating the delta rule during training. Tab. 1 shows the benefits of training with DeltaDEQ. For both Fourier and Siren variants of DEQ instantiations, applying the delta rule with training delta threshold $\tau = 1e-4$ reduces the training FLOPs in the forward pass for both phantom gradient and implicit function theorem used for the backward pass. Notice that the training FLOPs reduction level is obtained at the circumstances of **fixed-point reuse** (Sec. 4.3) for each training epoch and **only one fixed-point iteration** is executed except for the first epoch. This approach [32] has already significantly reduced the FLOPs required during the training forward pass to an asymptotically constant level [32]. Our method is orthogonal to this technique and can be implemented in conjunction with it. Furthermore, a single fixed-point iteration often does not suffice to accurately approximate $z^*$ for other complicated tasks, even when the fixed-point is reused. By applying our method, a greater reduction in training computations can be achieved.

| Training Method | Phantom Gradient [26] | | Implicit Function Theorem [2] | |
| --- | --- | --- | --- | --- |
| | Tr. FLOPs* | PSNR↑ | Tr. FLOPs* | PSNR↑ |
| DEQ-Fourier [32] | 140 | 31.31 | 278 | 33.22 |
| **DeltaDEQ-Fourier** | 110 (-21%) | 31.06 | 212 (-24%) | 32.98 |
| DEQ-Siren [32] | 140 | 29.94 | 278 | 34.96 |
| **DeltaDEQ-Siren** | 118 (-16%) | 29.64 | 190 (-32%) | 34.99 |

Table 1: Comparison of PSNR and training FLOPs w/o vs. **w** the delta rule during training. For reference, with same hyperparameters, the original Fourier and Siren (non-DEQ) networks recorded PSNRs of 30.06 and 33.31, respectively. All FLOPs values are presented in Tera-FLOPs (1e12). *This table only includes FLOPs for the forward pass of training; the computation cost of backward pass is independent of the forward pass.

## 5.2 Optical flow estimation

In this section, we study the application of DeltaDEQ on the classical computer vision task of optical flow (OF) estimation [16, 18, 31, 39, 53, 58]. OF estimation is a task for predicting the pixel displacement between two frames. Modern OF estimation is dominated by learning-based approaches and one of the representatives is the RAFT [53] network, which iteratively refines the OF estimation with a recurrent structure. In the work [4] the authors formulated the recurrent iterative structure of

RAFT as a DEQ block which consumes 4 to 6 times less training memory than the original RAFT with BPTT and achieved SOTA on the MPI Sintel [8] and KITTI 2015 [24] optical flow datasets.

**DEQ-RAFT architecture** We use the RAFT [53] network as the architectural backbone and the DEQ flow estimator version [4] (DEQ-RAFT) as our starting point for the conversion to DeltaDEQ. RAFT consists of mainly two parts: (1) the correlation and context encoders (feature extraction stage) and (2) the update block (iterative flow refinement stage). Although the feature extraction stage is also compatible with the delta rule and can be used to exploit the temporal correlation between the input frames, it is not the focus of this study since our main contribution is to accelerate the iterative computation in DEQ and other models. Thus, we omit the FLOPs of the first stage in results.

Assuming the feature extraction stage gives context embedding $q$ and correlation tensor $C$, the flow refinement stage conducts the fixed-point iteration simultaneously in two parts [4, 53]:

$$h^{i+1} = \mathcal{H}(h^i, o^i, q, C), \quad o^{i+1} = \mathcal{F}(h^{i+1}, o^i, q, C), \tag{12}$$

where $h$ is the hidden representation and $o$ is the optical flow estimation. Putting these notations into the fixed-point iteration framework in Eq.1, the extracted features $q, C$ are the input $x = (q, C)$, the iterative functions $\mathcal{H}, \mathcal{F}$ are the function $f_\theta$ and the hidden states and the optical flow make the fixed-point iteration $z^i = (h^i, o^i)$. Notice $i$ marks the fixed-point iteration step and all the terms are **input time** $t$ dependent when we move to process the next frame pair. Like in RAFT the optical flow refinement module is implemented with ConvGRU [5] for hidden states $h$ updates and other convolutional layers for the optical flow $o$ updates:

$$c^i = \text{Conv}([q, o^i, \text{Corr}(o^i + C)]), \; h^{i+1} = \text{ConvGRU}(h^i, [c^i, q]), \; o^{i+1} = o^i + \text{Conv}(h^{i+1}) \tag{13}$$

where Corr is the correlation lookup. Design details of DEQ-RAFT are in Fig. 10 and A.6.2.

**DeltaDEQ conversion of DEQ-RAFT** We convert the layers in the update block (optical flow refinement stage, Fig. 10) of the DEQ-RAFT (DEQ in Tab. 2 for simplicity). We use pretrained models provided in [4] and show even **without any fine-tuning**, we can achieve good computation reduction without hurting the task accuracy. We also observed the heterogeneous convergence phenomenon in the DEQ-RAFT network (Fig. 9) and during the convergence of fixed-point iteration, the delta activation sparsity also increases (Fig. 8), indicating the DEQ-RAFT network is suitable for applying the delta rule. DeltaDEQ conversion, pertaining and fine-tuning settings are given in A.6.2.

|   | Method | inf. $\tau$ | Sintel (train)↓ Clean | Final | $\Delta$ Sp. | FLOPs per pair | KITTI (train)↓ AEPE | F1-all | $\Delta$ Sp. | FLOPs per pair |
|---|--------|-----|-------|-------|-----|------|------|------|-----|------|
|   | PWC-Net [49] | - | 2.55 | 3.93 | - | - | 10.39 | 28.5 | - | - |
|   | VCN [57] | - | 2.21 | 3.68 | - | - | 8.36 | 25.1 | - | - |
|   | GMA [33] | - | 1.30 | 2.74 | - | - | 4.69 | 17.10 | - | - |
|   | RAFT [53] | - | 1.43 | 2.71 | - | - | 5.04 | 17.40 | - | - |
|   | DEQ(picard) [4] | - | 1.26 | 2.51 | - | 782G | 3.73 | 13.42 | - | 814G |
| 60 iters | DeltaDEQ | 0.0 | 1.22 | 2.55 | 0.52* | 394G | 3.77 | 13.47 | 0.54* | 396G |
| | DeltaDEQ | 0.001 | 1.24 | 2.52 | 0.69 | 264G | 3.79 | 13.49 | 0.69 | 274G |
| | DeltaDEQ | 0.005 | 1.48 | 2.76 | 0.80 | 177G | 4.00 | 14.04 | 0.80 | 183G |
| | DeltaDEQ | 0.01 | 2.08 | 3.32 | 0.83 | 147G | 4.33 | 15.85 | 0.84 | 150G |
| early stop | DEQ(picard) | - | 1.27 | 2.51 | - | 520G | 3.73 | 13.45 | - | 621G |
| | DeltaDEQ | 0.0 | 1.22 | 2.55 | 0.53* | 225G | 3.78 | 13.54 | 0.54* | 260G |
| | DeltaDEQ | 0.001 | 1.23 | 2.51 | 0.61 | 186G | 3.80 | 13.55 | 0.63 | 213G |
| | **DeltaDEQ** | 0.005 | 1.31 | 2.63 | 0.74 | **141G** | 3.90 | 13.75 | 0.74 | **152G** |
| | DeltaDEQ | 0.01 | 2.04 | 3.31 | 0.83 | 147G | 4.19 | 14.73 | 0.82 | 140G |
| | **DeltaDEQ-ft** | 0.005 | 1.25 | 2.59 | 0.73 | **127G** | 3.86 | 13.64 | 0.74 | **144G** |

Table 2: * Even for $\tau = 0.0$ the sparsity level is already 0.53. This high level of sparsity is in favor of our method since it is not exploited in the vanilla DEQ. For DEQ and DeltaDEQ models, the default forward method is KM with $\alpha = 0.9$. Notice with early stopping the total FLOPs could be lower even when the $\Delta$Sp. is lower since a different number of global fixed-point iterations could be executed.

In Tab. 2 we show the results of DeltaDEQ for the OF task in comparison with the original DEQ method [4] and other methods. The models were all pretrained on the FlyingChair [16] and FlyingThings3D [40] datasets and tested on the training splits of Sintel [8] and KITTI [24] datasets,

which is a common evaluation approach for the zero-shot generalization in OF. We report average end point error (AEPE) and F1-all (%) scores. We also report the delta activation sparsity $\Delta$Sp. which calculates $\frac{\text{\# delta activations}}{\text{\# total activations}}$ across all the iterations executed in the update block. We can see that applying the delta rule substantially reduced the averaged FLOPs needed in general. When using a fixed schedule of 60 iterations to process every input pair, the DeltaDEQ with $\tau = 0.001$ could save 77% and 66% of FLOPs on Sintel (train) and KITTI (train) in theory without substantially hurting the task accuracy. It is worth mentioning that even when the threshold is set to 0, we still got over 50% of $\Delta$Sp.. This is due to the fact of usage of ReLU activations. In combination with early stopping with relative distance criteria and tolerance of 0.001, we further reduce the average FLOPs to below 20% of originally needed. We also found that fine-tune the model (DeltaDEQ-ft) with $\tau = 0.005$ could improve the task accuracy while reducing the FLOPs and accelerating the convergence.

## 6   Related work

**Delta activation sparsity**  For accelerating network inference, one can exploit the temporal correlation, which naturally arises in time-dependent data such as audio and video. Most related to our work is that of [43] where the authors proposed to use a delta update rule, which zeros out small hidden states change between two input timesteps to skip computation. Later works realized this structured delta activation sparsity on custom hardware [12, 14, 20–22] and turned the theoretical FLOPs reduction into runtime acceleration. Another line of work studies similar temporal-dependent delta activation sparsity but with CNNs. In [11, 29, 29, 44, 45] the authors propose to exploit the linearity of the convolution operator and apply the delta rule to skip zero delta activities for processing consecutive video frames.

Several challenges arise with these approaches. To implement the delta rule in CNN architectures, it is necessary to store the intermediate feature maps of all convolution layers for computing the delta activations. If the network incorporates many layers and has low parameter efficiency, this will lead to substantial memory consumption. Although DeltaDEQ also requires storing feature maps, its property of high parameter efficiency could alleviate this issue by requiring fewer feature maps to be stored. Secondly, there is no guarantee that input data from consecutive time steps will produce temporally correlated features. Consequently, employing the delta rule might result in more overhead, offering no computational advantages. However, due to the fixed-point nature of DeltaDEQ, it is guaranteed to have a certain degree of sparsity from the delta activations between two FP iterations.

**Deep equilibrium models**  Deep equilibrium (DEQ) models [2, 25] formulate stacking multiple layers of conventional network layers as a fixed-point solving problem of a single layer or a block of layers transformation. This could result in a smaller memory footprint and better parameter efficiency. DEQ models have also shown state-of-art task performance in various practical applications including object recognition or detection [3, 17], optical flow estimation [4], face landmark detection [41], implicit neural representation [32] and many others [10, 46]. One of the major drawbacks of DEQ models is its computation cost: for finding the fixed point, many fixed-point iterations or iterations in root-solving are usually required and every iteration requires one inference of the DEQ layer.

**Differences to other network acceleration algorithms**  In comparison to layer skipping methods [9, 27, 29] which skip certain layers or halts the recurrence early in inference, our method exploits a finer-grained delta activation sparsity while the other methods can only skip an entire layer or recurrent step. Another line of works dynamically prunes activation, including channel selection [23] for CNNs and patch selection for transformer architectures [52, 55]. These works usually set certain channels or tokens of the activations to zeros or discard them so the network enjoys activation sparsity. Our method allows the activations to maintain non-zero values, while still generating delta activation sparsity, which could be exploited for computation saving. The non-zero activations could potentially make the network more expressive.

## 7   Conclusion

In this work, we introduce DeltaDEQ, a method designed to enhance computational efficiency for implicit models represented by deep equilibrium models. This method is inspired by our observation of the heterogeneous convergence phenomenon prevalent in implicit neural networks, where different dimensions of DEQ hidden states converge at varying speeds. DeltaDEQ leverages this disparity by

calculating the delta activation between consecutive fixed-point iterations and utilizing the activation's sparsity to omit unnecessary computations for the already converged dimensions. We have tested DeltaDEQ across tasks involving implicit neural representation and optical flow, employing both RNN and CNN architectures. Our findings confirm that DeltaDEQ maintains accuracy while reducing computational demands across these network types. We provide detailed theoretical FLOPs reductions from our empirical research. This technique has the potential societal impact to decrease energy usage significantly in DEQ models [2, 4, 10, 17, 41, 46] and other iterative methods, including those used in iterative refinement [53] and emerging diffusion models [30, 46]. However, translating these theoretical savings into actual wall-clock inference latency reductions often demands carefully designed implementation [15] and special hardware [1].

## Acknowledgments and Disclosure of Funding

This work was partially funded by the European Union's Horizon 2020 research, innovation programme under grand agreement No 899287 and the Swiss National Science Foundation CA-DNNEdge project (208227).

## Footnotes

[1]In the original works [2, 4, 26] DEQ refers to the models trained with implicit function theorem (IFT) or approximation based on IFT. In this work we relax the definition of DEQ to any network that models a fixed-point equation and the forward pass is solved by fixed-point iteration or other root-finder.

## References

[1] Alessandro Aimar, Hesham Mostafa, Enrico Calabrese, Antonio Rios-Navarro, Ricardo Tapiador-Morales, Iulia-Alexandra Lungu, Moritz B. Milde, Federico Corradi, Alejandro Linares-Barranco, Shih-Chii Liu, and Tobi Delbruck. Nullhop: A flexible convolutional neural network accelerator based on sparse representations of feature maps. *IEEE Transactions on Neural Networks and Learning Systems*, 30(3): 644–656, 2019. doi: 10.1109/TNNLS.2018.2852335. (Cited on pages 4, 5, and 10.)

[2] Shaojie Bai, J. Zico Kolter, and Vladlen Koltun. Deep equilibrium models. In *Advances in Neural Information Processing Systems*, volume 32. Curran Associates, Inc., 2019. URL https://proceedings.neurips.cc/paper_files/paper/2019/file/01386bd6d8e091c2ab4c7c7de644d37b-Paper.pdf. (Cited on pages 1, 2, 5, 6, 7, 9, 10, and 15.)

[3] Shaojie Bai, Vladlen Koltun, and J. Zico Kolter. Multiscale deep equilibrium models. In *Advances in Neural Information Processing Systems (NeurIPS)*, 2020. (Cited on pages 1 and 9.)

[4] Shaojie Bai, Zhengyang Geng, Yash Savani, and J. Zico Kolter. Deep equilibrium optical flow estimation. In *2022 IEEE/CVF Conference on Computer Vision and Pattern Recognition (CVPR)*, pp. 610–620, Los Alamitos, CA, USA, jun 2022. IEEE Computer Society. doi: 10.1109/CVPR52688.2022.00070. URL https://doi.ieeecomputersociety.org/10.1109/CVPR52688.2022.00070. (Cited on pages 1, 2, 5, 7, 8, 9, 10, 17, 19, and 20.)

[5] Nicolas Ballas, Li Yao, Chris Pal, and Aaron Courville. Delving deeper into convolutional networks for learning video representations. *arXiv preprint arXiv:1511.06432*, 2015. (Cited on page 8.)

[6] Stefan Banach. Sur les opérations dans les ensembles abstraits et leur application aux équations intégrales. *Fundamenta mathematicae*, 3(1):133–181, 1922. (Cited on page 3.)

[7] Charles George Broyden. A class of methods for solving nonlinear simultaneous equations. *Mathematics of Computation*, 19:577–593, 1965. URL https://api.semanticscholar.org/CorpusID:2802972. (Cited on pages 1, 2, 5, and 15.)

[8] Daniel J Butler, Jonas Wulff, Garrett B Stanley, and Michael J Black. A naturalistic open source movie for optical flow evaluation. In *Computer Vision–ECCV 2012: 12th European Conference on Computer Vision, Florence, Italy, October 7-13, 2012, Proceedings, Part VI 12*, pp. 611–625. Springer, 2012. (Cited on pages 8 and 19.)

[9] Víctor Campos, Brendan Jou, Xavier Giró-i-Nieto, Jordi Torres, and Shih-Fu Chang. Skip RNN: learning to skip state updates in recurrent neural networks. In *6th International Conference on Learning Representations, ICLR 2018, Vancouver, BC, Canada, April 30 - May 3, 2018, Conference Track Proceedings*. OpenReview.net, 2018. URL https://openreview.net/forum?id=HkwVAXyCW. (Cited on page 9.)

[10] Qi Chen, Yifei Wang, Zhengyang Geng, Yisen Wang, Jiansheng Yang, and Zhouchen Lin. Equilibrium image denoising with implicit differentiation. *IEEE Transactions on Image Processing*, 32:1868–1881, 2023. doi: 10.1109/TIP.2023.3255104. (Cited on pages 9 and 10.)

[11] Qinyu Chen, Zuowen Wang, Shih-Chii Liu, and Chang Gao. 3et: Efficient event-based eye tracking using a change-based convlstm network. In *2023 IEEE Biomedical Circuits and Systems Conference (BioCAS)*, pp. 1–5, 2023. doi: 10.1109/BioCAS58349.2023.10389062. (Cited on page 9.)

[12] Qinyu Chen, Kwantae Kim, Chang Gao, Sheng Zhou, Taekwang Jang, Tobi Delbruck, and Shih-Chii Liu. A 65nm 36nJ/Decision bio-inspired temporal-sparsity-aware digital keyword spotting IC with 0.6 V near-threshold SRAM. *arXiv preprint arXiv:2405.03905*, 2024. (Cited on page 9.)

[13] Ricky T. Q. Chen, Yulia Rubanova, Jesse Bettencourt, and David K Duvenaud. Neural ordinary differential equations. In *Advances in Neural Information Processing Systems*, volume 31. Curran Associates, Inc., 2018. URL `https://proceedings.neurips.cc/paper_files/paper/2018/file/69386f6bb1dfed68692a24c8686939b9-Paper.pdf`. (Cited on page 1.)

[14] Xi Chen, Chang Gao, Zuowen Wang, Longbiao Cheng, Sheng Zhou, Shih-Chii Liu, and Tobi Delbruck. Exploiting symmetric temporally sparse BPTT for efficient RNN training. *Proceedings of the AAAI Conference on Artificial Intelligence*, 38(10):11399–11406, Mar. 2024. doi: 10.1609/aaai.v38i10.29020. URL `https://ojs.aaai.org/index.php/AAAI/article/view/29020`. (Cited on page 9.)

[15] Spconv Contributors. SpConv: Spatially sparse convolution library. `https://github.com/traveller59/spconv`, 2022. (Cited on pages 4, 5, and 10.)

[16] Alexey Dosovitskiy, Philipp Fischer, Eddy Ilg, Philip Hausser, Caner Hazirbas, Vladimir Golkov, Patrick Van Der Smagt, Daniel Cremers, and Thomas Brox. Flownet: Learning optical flow with convolutional networks. In *Proceedings of the IEEE international conference on computer vision*, pp. 2758–2766, 2015. (Cited on pages 7, 8, and 20.)

[17] Can Ufuk Ertenli, Emre Akbaş, and Ramazan Gökberk Cinbiş. Streaming multiscale deep equilibrium models. In *17th European Conference on Computer Vision, ECCV 2022*, 2022. URL `https://www.ecva.net/papers/eccv_2022/papers_ECCV/papers/136710189-supp.pdf`. (Cited on pages 1, 9, and 10.)

[18] David Fleet and Yair Weiss. Optical flow estimation. In *Handbook of mathematical models in computer vision*, pp. 237–257. Springer, 2006. (Cited on page 7.)

[19] Samy Wu Fung, Howard Heaton, Qiuwei Li, Daniel McKenzie, Stanley Osher, and Wotao Yin. JFB: Jacobian-free backpropagation for implicit models. *Proceedings of the AAAI Conference on Artificial Intelligence*, 2022. (Cited on page 6.)

[20] Chang Gao, Daniel Neil, Enea Ceolini, Shih-Chii Liu, and Tobi Delbruck. DeltaRNN: A power-efficient recurrent neural network accelerator. In *Proceedings of the 2018 ACM/SIGDA International Symposium on Field-Programmable Gate Arrays*, FPGA '18, pp. 21–30, 2018. doi: 10.1145/3174243.3174261. (Cited on pages 5 and 9.)

[21] Chang Gao, Stefan Braun, Ilya Kiselev, Jithendar Anumula, Tobi Delbruck, and Shih-Chii Liu. Real-time speech recognition for IoT purpose using a delta recurrent neural network accelerator. In *2019 IEEE International Symposium on Circuits and Systems (ISCAS)*, pp. 1–5, 2019. doi: 10.1109/ISCAS.2019.8702290. (Not cited.)

[22] Chang Gao, Tobi Delbruck, and Shih-Chii Liu. Spartus: A 9.4 Top/s FPGA-based LSTM accelerator exploiting spatio-temporal sparsity. *IEEE Transactions on Neural Networks and Learning Systems*, 35(1): 1098–1112, 2024. doi: 10.1109/TNNLS.2022.3180209. (Cited on page 9.)

[23] Xitong Gao, Yiren Zhao, Łukasz Dudziak, Robert Mullins, and Cheng zhong Xu. Dynamic channel pruning: Feature boosting and suppression. In *International Conference on Learning Representations*, 2019. URL `https://openreview.net/forum?id=BJxh2j0qYm`. (Cited on page 9.)

[24] Andreas Geiger, Philip Lenz, Christoph Stiller, and Raquel Urtasun. Vision meets robotics: The kitti dataset. *The International Journal of Robotics Research*, 32(11):1231–1237, 2013. (Cited on page 8.)

[25] Zhengyang Geng and J. Zico Kolter. TorchDEQ: A library for deep equilibrium models. `https://github.com/locuslab/torchdeq`, 2023. (Cited on page 9.)

[26] Zhengyang Geng, Xin-Yu Zhang, Shaojie Bai, Yisen Wang, and Zhouchen Lin. On training implicit models. *Advances in Neural Information Processing Systems*, 34:24247–24260, 2021. (Cited on pages 2, 6, 7, and 20.)

[27] Alex Graves. Adaptive computation time for recurrent neural networks, 2017. URL `https://arxiv.org/abs/1603.08983`. (Cited on page 9.)

[28] Swaminathan Gurumurthy, Shaojie Bai, Zachary Manchester, and J Zico Kolter. Joint inference and input optimization in equilibrium networks. In *Thirty-Fifth Conference on Neural Information Processing Systems*, 2021. URL `https://openreview.net/forum?id=RgHOgGH9B64`. (Cited on page 6.)

[29] Amirhossein Habibian, Davide Abati, Taco S. Cohen, and Babak Ehteshami Bejnordi. Skip-convolutions for efficient video processing. In *2021 IEEE/CVF Conference on Computer Vision and Pattern Recognition (CVPR)*, pp. 2694–2703, Los Alamitos, CA, USA, jun 2021. IEEE Computer Society. doi: 10.1109/CVPR46437.2021.00272. URL `https://doi.ieeecomputersociety.org/10.1109/CVPR46437.2021.00272`. (Cited on page 9.)

[30] Jonathan Ho, Ajay Jain, and Pieter Abbeel. Denoising diffusion probabilistic models. *Advances in neural information processing systems*, 33:6840–6851, 2020. (Cited on page 10.)

[31] Berthold KP Horn and Brian G Schunck. Determining optical flow. *Artificial intelligence*, 17(1-3):185–203, 1981. (Cited on page 7.)

[32] Zhichun Huang, Shaojie Bai, and J. Zico Kolter. $(implicit)^2$: Implicit layers for implicit representations. In *Advances in Neural Information Processing Systems (NeurIPS)*, 2021. (Cited on pages 1, 2, 5, 6, 7, 9, and 17.)

[33] Shihao Jiang, Dylan Campbell, Yao Lu, Hongdong Li, and Richard Hartley. Learning to estimate hidden motions with global motion aggregation. In *Proceedings of the IEEE/CVF international conference on computer vision*, pp. 9772–9781, 2021. (Cited on page 8.)

[34] Diederik P. Kingma and Jimmy Ba. Adam: A method for stochastic optimization. In *3rd International Conference on Learning Representations, ICLR 2015, San Diego, CA, USA, May 7-9, 2015, Conference Track Proceedings*, 2015. URL `http://arxiv.org/abs/1412.6980`. (Cited on page 6.)

[35] Yuichiro Koyama, Naoki Murata, Stefan Uhlich, Giorgio Fabbro, Shusuke Takahashi, and Yuki Mitsufuji. Music source separation with deep equilibrium models. In *ICASSP 2022-2022 IEEE International Conference on Acoustics, Speech and Signal Processing (ICASSP)*, pp. 296–300. IEEE, 2022. (Cited on page 1.)

[36] Steven George Krantz and Harold R Parks. *The implicit function theorem: history, theory, and applications*. Springer Science & Business Media, 2002. (Cited on page 6.)

[37] Renate Krause, Matthew Cook, Sepp Kollmorgen, Valerio Mante, and Giacomo Indiveri. Operative dimensions in unconstrained connectivity of recurrent neural networks. In *Advances in Neural Information Processing Systems*, volume 35, pp. 17073–17085. Curran Associates, Inc., 2022. URL `https://proceedings.neurips.cc/paper_files/paper/2022/file/6cdd4ce9330025967dd1ed0bed3010f5-Paper-Conference.pdf`. (Cited on page 3.)

[38] Ilya Loshchilov and Frank Hutter. SGDR: Stochastic gradient descent with warm restarts. *arXiv preprint arXiv:1608.03983*, 2016. (Cited on page 6.)

[39] Bruce D Lucas and Takeo Kanade. An iterative image registration technique with an application to stereo vision. In *IJCAI'81: 7th international joint conference on Artificial intelligence*, volume 2, pp. 674–679, 1981. (Cited on page 7.)

[40] Nikolaus Mayer, Eddy Ilg, Philip Hausser, Philipp Fischer, Daniel Cremers, Alexey Dosovitskiy, and Thomas Brox. A large dataset to train convolutional networks for disparity, optical flow, and scene flow estimation. In *Proceedings of the IEEE conference on computer vision and pattern recognition*, pp. 4040–4048, 2016. (Cited on pages 8 and 20.)

[41] Paul Micaelli, Arash Vahdat, Hongxu Yin, Jan Kautz, and Pavlo Molchanov. Recurrence without recurrence: Stable video landmark detection with deep equilibrium models. In *2023 IEEE/CVF Conference on Computer Vision and Pattern Recognition (CVPR)*, pp. 22814–22825, Los Alamitos, CA, USA, jun 2023. IEEE Computer Society. doi: 10.1109/CVPR52729.2023.02185. URL `https://doi.ieeecomputersociety.org/10.1109/CVPR52729.2023.02185`. (Cited on pages 1, 2, 5, 9, and 10.)

[42] Takeru Miyato, Toshiki Kataoka, Masanori Koyama, and Yuichi Yoshida. Spectral normalization for generative adversarial networks. *arXiv preprint arXiv:1802.05957*, 2018. (Cited on page 6.)

[43] Daniel Neil, Junhaeng Lee, Tobi Delbrück, and Shih-Chii Liu. Delta networks for optimized recurrent network computation. In *Proceedings of the 34th International Conference on Machine Learning, ICML 2017*, pp. 2584–2593, 2017. (Cited on pages 5 and 9.)

[44] Mathias Parger, Chengcheng Tang, Christopher Twigg, Cem Keskin, Robert Wang, and Markus Steinberger. DeltaCNN: End-to-end CNN inference of sparse frame differences in videos. In *2022 IEEE/CVF Conference on Computer Vision and Pattern Recognition (CVPR)*, pp. 12487–12496, Los Alamitos, CA, USA, jun 2022. IEEE Computer Society. doi: 10.1109/CVPR52688.2022.01217. URL `https://doi.ieeecomputersociety.org/10.1109/CVPR52688.2022.01217`. (Cited on pages 4 and 9.)

[45] Mathias Parger, Chengcheng Tang, Thomas Neff, Christopher D. Twigg Twigg, Cem Keskin, Robert Wang, and Markus Steinberger. MotionDeltaCNN: Sparse CNN inference of frame differences in moving camera videos with spherical buffers and padded convolutions. In *2023 IEEE/CVF International Conference on Computer Vision (ICCV)*, pp. 17246–17255, Los Alamitos, CA, USA, oct 2023. IEEE Computer Society. doi: 10.1109/ICCV51070.2023.01586. URL `https://doi.ieeecomputersociety.org/10.1109/ICCV51070.2023.01586`. (Cited on pages 4 and 9.)

[46] Ashwini Pokle, Zhengyang Geng, and J. Zico Kolter. Deep equilibrium approaches to diffusion models. In *Advances in Neural Information Processing Systems*, volume 35, pp. 37975–37990. Curran Associates, Inc., 2022. URL `https://proceedings.neurips.cc/paper_files/paper/2022/file/f7f47a73d631c0410cbc2748a8015241-Paper-Conference.pdf`. (Cited on pages 1, 9, 10, and 15.)

[47] Jack Sherman and Winifred J Morrison. Adjustment of an inverse matrix corresponding to a change in one element of a given matrix. *The Annals of Mathematical Statistics*, 21(1):124–127, 1950. (Cited on page 15.)

[48] Vincent Sitzmann, Julien Martel, Alexander Bergman, David Lindell, and Gordon Wetzstein. Implicit neural representations with periodic activation functions. *Advances in neural information processing systems*, 33:7462–7473, 2020. (Cited on page 6.)

[49] Deqing Sun, Xiaodong Yang, Ming-Yu Liu, and Jan Kautz. PWC-Net: CNNs for optical flow using pyramid, warping, and cost volume. In *Proceedings of the IEEE conference on computer vision and pattern recognition*, pp. 8934–8943, 2018. (Cited on page 8.)

[50] David Sussillo and Omri Barak. Opening the black box: Low-dimensional dynamics in high-dimensional recurrent neural networks. *Neural Computation*, 25(3):626–649, March 2013. ISSN 0899-7667. doi: 10.1162/NECO_a_00409. URL `https://doi.org/10.1162/NECO_a_00409`. (Cited on page 3.)

[51] Matthew Tancik, Pratul Srinivasan, Ben Mildenhall, Sara Fridovich-Keil, Nithin Raghavan, Utkarsh Singhal, Ravi Ramamoorthi, Jonathan Barron, and Ren Ng. Fourier features let networks learn high frequency functions in low dimensional domains. *Advances in neural information processing systems*, 33: 7537–7547, 2020. (Cited on page 6.)

[52] Yehui Tang, Kai Han, Yunhe Wang, Chang Xu, Jianyuan Guo, Chao Xu, and Dacheng Tao. Patch slimming for efficient vision transformers. In *Proceedings of the IEEE/CVF Conference on Computer Vision and Pattern Recognition*, pp. 12165–12174, 2022. (Cited on page 9.)

[53] Zachary Teed and Jia Deng. Raft: Recurrent all-pairs field transforms for optical flow. In *Computer Vision–ECCV 2020: 16th European Conference, Glasgow, UK, August 23–28, 2020, Proceedings, Part II 16*, pp. 402–419. Springer, 2020. (Cited on pages 7, 8, 10, and 20.)

[54] Homer F. Walker and Peng Ni. Anderson acceleration for fixed-point iterations. *SIAM Journal on Numerical Analysis*, 49(4):1715–1735, 2011. doi: 10.1137/10078356X. URL `https://doi.org/10.1137/10078356X`. (Cited on pages 1, 2, and 5.)

[55] Zuowen Wang, Yuhuang Hu, and Shih-Chii Liu. Exploiting spatial sparsity for event cameras with visual transformers. In *2022 IEEE International Conference on Image Processing (ICIP)*, pp. 411–415, 2022. doi: 10.1109/ICIP46576.2022.9897432. (Cited on page 9.)

[56] Zuowen Wang, Longbiao Cheng, Joachim Ott, Pehuen Moure, and Shih-Chii Liu. Bio-inspired parameter reuse: Exploiting inter-frame representation similarity with recurrence for accelerating temporal visual processing. In *Proceedings of UniReps: the First Workshop on Unifying Representations in Neural Models*, volume 243 of *Proceedings of Machine Learning Research (PMLR)*, pp. 209–222. PMLR, 15 Dec 2024. URL `https://proceedings.mlr.press/v243/wang24a.html`. (Cited on pages 2 and 5.)

[57] Gengshan Yang and Deva Ramanan. Volumetric correspondence networks for optical flow. *Advances in neural information processing systems*, 32, 2019. (Cited on page 8.)

[58] Mingliang Zhai, Xuezhi Xiang, Ning Lv, and Xiangdong Kong. Optical flow and scene flow estimation: A survey. *Pattern Recognition*, 114:107861, 2021. (Cited on page 7.)

# A Appendices

## A.1 PCA details

In this section we describe the details for the methods used in Fig. 1(b) and (c).

For Fig. 1(b) we randomly selected 5 neighboring $x$ in $[-2\pi, 2\pi]$ from the training set as model inputs. The inference fixed-point iteration was 15 steps, same as in training. Thus, this results in $5 \times 15 = 75$ hidden states, each of dimension 20. We conduct PCA on the $\mathbb{R}^{75 \times 20}$ hidden states matrix and plot the first 2 sorted principle components and visualize the hidden states in 2D. The colors of the dots are selected so the hidden state from the $i$-th iteration has the same color. We demonstrate with Fig. 1(b) that if input points are close in the input space, their distances in the hidden states are also close, which justifies the acceleration effect of fixed-point reuse.

For Fig. 1(c) we infer the model on 200 data samples in $[-2\pi, 2\pi]$ and store the $200 \times 15 = 3000$ hidden states. The resulting hidden state matrix is of dimension $\mathbb{R}^{7500 \times 20}$. We conduct PCA on this matrix and plot the cumulative explained variance of the top 8 principal components.

## A.2 Complete delta formulation for a convolution layer

In this section, we give the complete formulation for a convolution layer in a DeltaDEQ when applying the delta rule. For the input features to a layer in a DeltaDEQ block at iteration $i$ is $I_t^i$ and the output is $O_t^i$:

$$O_t^i = \sigma(\mathcal{K} * I_t^i) = \sigma(\mathcal{K} * (I_t^i - I_t^{i-1} + I_t^{i-1})) \tag{14}$$

$$= \sigma(\mathcal{K} * (I_t^i - I_t^{i-1}) + \mathcal{K} * I_t^{i-1})) \tag{15}$$

$$\Delta I_t^i := \begin{cases} I_t^i - I_t^{i-1} & \text{if } |I_t^i - I_t^{i-1}| \geq \tau \\ 0 & \text{otherwise} \end{cases} \quad \textbf{(Delta Rule}, \text{applied elementwise)} \tag{16}$$

$$O_t^i \overset{(16)}{\approx} \sigma(\underbrace{\mathcal{K} * \Delta I_t^i}_{\text{sparse}} + \underbrace{\mathcal{K} * I_t^{i-1}}_{\text{cached in } C_z}) \tag{17}$$

$$C_z \leftarrow \mathcal{K} * \Delta I_t^i + C_z \quad \text{(update cache, already calculated in 17)} \tag{18}$$

where $*$ denotes the convolution operator and $\mathcal{K}$ represents the convolution kernel. The illustration is given in Fig. 2. Here we provide the pseudo-code for a convolution layer with delta rule in DeltaDEQ:

---

**Algorithm 1:** An convolution layer with delta rule Eq. 17

---

**Input** : Output feature map $I_t^i$ from the previous layer, could also be input data if this convolution layer is the first layer of the network. Convolution kernel $\mathcal{K}$. Delta threshold $\tau$.

**Cached** : Output feature map $I_t^{i-1}$. Linear operation results $C$

**Result:** Output of this convolution layer $O_t^i$. Updated linear operation results $C$

1 **if** $i == 0$ **then**
    /* First iteration, initialize the cache with convolution on the original input                                                                */
2    $C \leftarrow \mathcal{K} * I_t^0$
3 **else**
    /* Apply delta rule elementwise                                                                */
4    $\Delta I_t^i \leftarrow \begin{cases} I_t^i - I_t^{i-1} & \text{if } |I_t^i - I_t^{i-1}| \geq \tau \\ 0 & \text{otherwise} \end{cases}$
    /* Update the linear operator results to the cache                                                                */
5    $C \leftarrow C + \mathcal{K} * \Delta I_t^i$
6 **end**
    /* Apply the non-linearity function $\sigma$                                                                */
7 $O_t^i \leftarrow \sigma(C)$
    /* Move on to the next convolution layer, or if the DeltaDEQ block is finished processing, move on to the next iteration $i + 1$.                                                                */

---

### A.3 Applying delta rule with other fixed-point solving techniques

Not only can the delta rule be applied in fixed-point iteration such as in formulation 2, it can also be utilized in other solvers commonly used in other DEQ works. We take the Broyden method as a example. As described in the work [2], the Broyden's method [7] updates the hidden states as:

$$z_t^{i+1} = z_t^i - \alpha B g_\theta(z_t^i, x_t) \tag{19}$$

where $g_\theta(z_t^i, x_t) = f_\theta(z_t^i, x_t) - z_t^i$, $B$ is the Jacobian inverse and $\alpha$ is the step size. This update step requires computing a forward pass $f_\theta(z_t^i, x_t)$ which is same as the vanilla fixed-point iteration as in Eq. 2. Then the computing of this forward can be approximated and accelerated same as in Eq. 6 or Eq. 17. Moreover, the other computatinally heavy part of Broyden's method, which is the calculation of the inverse Jacobian could also potentaily be accelerated with delta rule. Considering the inverse Jacobian approximation method Sherman-Morrison formula [47] (omitting the timestep $t$ for simplicity):

$$B^{i+1} = B^i + \frac{\Delta z^{i+1} - B^i \Delta g_\theta^{i+1}}{\Delta z^{i+1} B^i \Delta g_\theta^{i+1}} (\Delta z^{i+1})^T B^i \tag{20}$$

where $\Delta z^{i+1} = z^{i+1} - z^i$ and $\Delta g_\theta^{i+1} = g_\theta(z^{i+1}, x) - g_\theta(z^i, x) = (f_\theta(z^{i+1} - z^{i+1}) - (f_\theta(z^i - z^i) = (f_\theta(z^{i+1} - f_\theta(z^i) - \Delta z^{i+1}$. So theoretically we could apply the delta rule on $\Delta z^{i+1}$ and $g_\theta^{i+1} - g_\theta^i$ in order to create sparse vectors in order to have sparse matrix vector multiplications in the Sherman-Morrison formula. We leave the experimental verification to future works.

### A.4 Feasibility of applying the delta rule in transformer-based DEQ networks

In this section, we discuss the compatibility of the delta rule with transformer-based DEQ networks, as the transformer architecture is widely used in various applications [46] so their DEQ version also has great potential. Here we provide the conceptual delta formulation for the two computationally costly components in a transformer architecture namely the fully connected layers and the self-attention module.

Computing the fully connected layers consists mainly of matrix-vector multiplications followed by nonlinearity $\sigma$, $\text{FFN}(x^i) = \sigma(Wx^i)$. Multiplication of matrix vectors $Wx^i$ can be formulated with the delta rule just like in the RNN-based DeltaDEQ in Eq.5.

The self-attention module $\text{Self-Attention}(Q, K, V) = \text{softmax}(QK^T/\sqrt{d})V$, where $Q^i = Z^i W_Q, K^i = Z^i W_K, V^i = Z^i W_V$, and $Z^i \in \mathcal{R}^{T \times d}$ is the state of the DEQ-transformer architecture, $T$ is the sequence length (equal to the number of tokens) and d is the dimension size. Taking the construction of $Q$ as an example, using the delta rule on the computation of $Q^i, K^i, V^i$ gives us:

$$Q^i = (Z^i - Z^{i-1} + Z^{i-1})W_Q \tag{21}$$
$$= (Z^i - Z^{i-1})W_Q + Z^{i-1}W_Q \tag{22}$$
$$= \Delta Z^i W_Q + C_Q \tag{23}$$

where the elements in $\Delta Z^i$ are set to corresponding elements in $(Z^i - Z^{i-1})$ if the absolute value of individual dimensions is greater than the delta threshold $\tau$, otherwise it is set to 0. Thus $\Delta Z^i W_Q$ could be a sparse matrix-matrix multiplication, depending on how $Z^i$ evolves. $C_Q$ is the cached results $C_Q$ which will be updated by $C_Q \leftarrow C_Q + \Delta Z^i W_Q$. Computing $K^i$ and $V^i$ follows the same pattern. The following operations such as activation function and layer normalization remain the same as a regular transformer model. The computation savings for the self-attention module come from the sparse matrix-matrix multiplication. Assuming the sparsity percentage of $\Delta Z$ is $o\%$ then the FLOPs for computing $Q^i$ is $o\% \cdot T \cdot d^2$ in comparison to $Td^2$.

### A.5 DeltaDEQ for INR: methods and computing platform details and additional results

#### A.5.1 Illustration for RNN type of DeltaDEQ

We show two steps of fixed-point iteration for the RNN type of DeltaDEQ in Fig. 5. An additional cache needs to be maintained to store the results of the linear components inside the activation function. Here we omit the delta rule on the input $x_t$ and focus on the part for hidden state $z$.

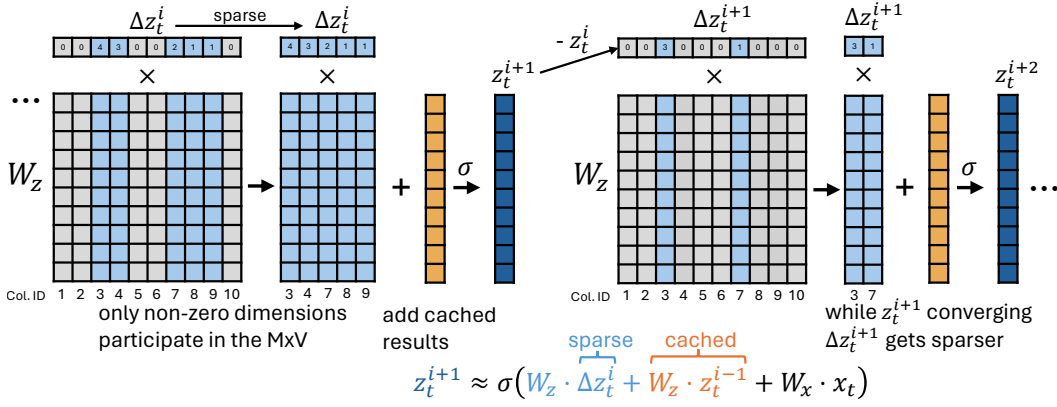

$$z_t^{i+1} \approx \sigma\left(W_z \cdot \Delta z_t^i + W_z \cdot z_t^{i-1} + W_x \cdot x_t\right)$$

Figure 5: Illustration of saving mechanisms of DeltaDEQ. Fig. Fully connected $W_z \cdot z_t^i$ type of computation skip. Entire columns of MACs at zero entries of $\Delta z_t^i$ can be skipped and the sparsity of $z_t^i$ grows with iteration $i$.

#### A.5.2 Additional results

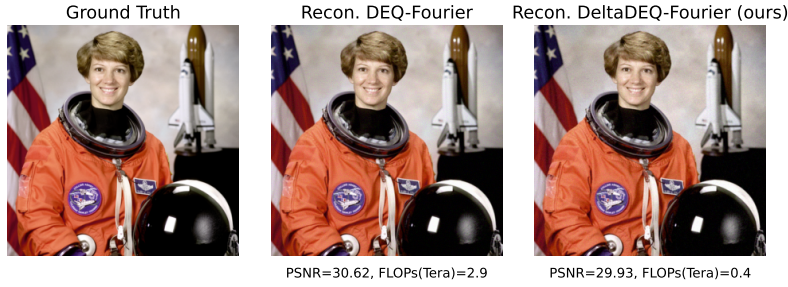

Figure 6: Original image of the astronaut and reconstructions with INR network. For DeltaDEQ-Fourier, $\tau = 0.05$.

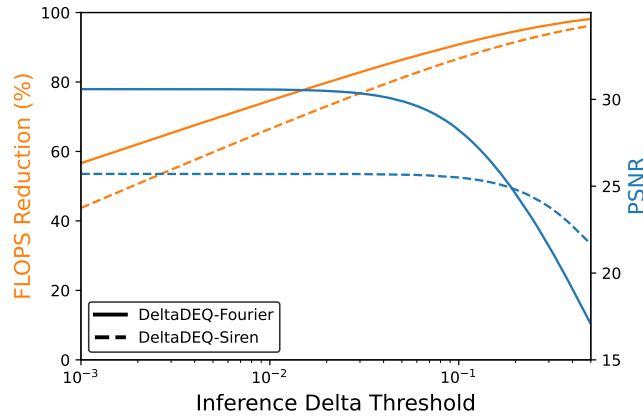

Figure 7: FLOPs reduction and task accuracy (PSNR) at different inference delta threshold on the astronaut image.

| Training Method | Phantom Gradient | | Implicit Function Theorem | |
|---|---|---|---|---|
| | Tr. FLOPs* | PSNR↑ | Tr. FLOPs* | PSNR↑ |
| DEQ-Fourier [32] | 140.8 | 29.90 | 278.4 | 32.04 |
| **DeltaDEQ-Fourier** | 112.6 | 29.60 | 215.5 | 31.66 |
| DEQ-Siren [32] | 140.8 | 25.67 | 378.5 | 30.01 |
| **DeltaDEQ-Siren** | 122.3 | 25.59 | 219.6 | 30.06 |

Table 3: Results on Astraunaut. Comparison of PSNR and training FLOPs w/o vs. **w** the delta rule during training. For reference, with same hyperparameters, the original Fourier and Siren (non-DEQ) networks recorded PSNRs of 30.06 and 33.31, respectively. All FLOPs values are presented in Tera-FLOPs (1e12). *This table only includes FLOPs for the forward pass of training; the computation cost of backward pass is independent of the forward pass.

### A.5.3 Delta rule formulation for DeltaDEQ-Siren and DeltaDEQ-Fourier

The formulations of DeltaDEQ-Siren and DeltaDEQ-Fourier are given as follows:

DeltaDEQ-Siren: $\qquad$ (24)

$$z_t^{i+1} = \text{Sin}\left(W(z_t^i - z_t^{i-1}) + Wz_t^{i-1} + W\text{Sin}(Vx) + Ux + b\right) \qquad (25)$$

$$\approx \text{Sin}\left(\underbrace{\underbrace{W\Delta z_t^i}_{\text{sparse}} + \underbrace{Wz_t^{i-1} + W\text{Sin}(Vx)}_{\text{cached in } C_z}}_{\text{update cache } C_z} + \underbrace{Ux + b}_{\text{cached in } C_x}\right) \qquad (26)$$

DeltaDEQ-Fourier: $\qquad$ (27)

$$z_t^{i+1} = (W(z_t^i - z_t^{i-1}) + Wz_t^{i-1} + Wg(x;V) + b) \circ g(x;U) \qquad (28)$$

$$\approx \left(\underbrace{\underbrace{W\Delta z_t^i}_{\text{sparse}} + \underbrace{Wz_t^{i-1} + Wg(x;V)}_{\text{cached in } C_z}}_{\text{update cache } C_z} + b\right) \circ \underbrace{g(x;U)}_{\text{cached in } C_x} \qquad (29)$$

$$\qquad (30)$$

### A.6 DeltaDEQ for optical flow

### A.6.1 Heterogeneous convergence for CNN-based DEQ

In Fig. 9 we present a convergence analysis for CNN-based DEQ model [4]. The model uses the RAFT architecture A.6.2 which is the state-of-the-art for optical flow estimation. We randomly selected 60 feature map pixel locations from the separable ConvGRU and plotted their evolution during the processing of 5 pairs of input images. The activation values are normalized and sorted according to their average. The basic fixed-point iteration namely the Picard iteration is applied. From the figure, we can see that certain activation values converge faster than others and some activations oscillate. Fig. 8 shows the sparsity level of the delta feature maps (original DEQ-flow applying the delta rule with threshold 0.005). We can see that different components tend to reach high delta sparsity at different speeds on average. This also reflects the heterogeneous convergence phenomenon also exists in the CNN-based implicit models.

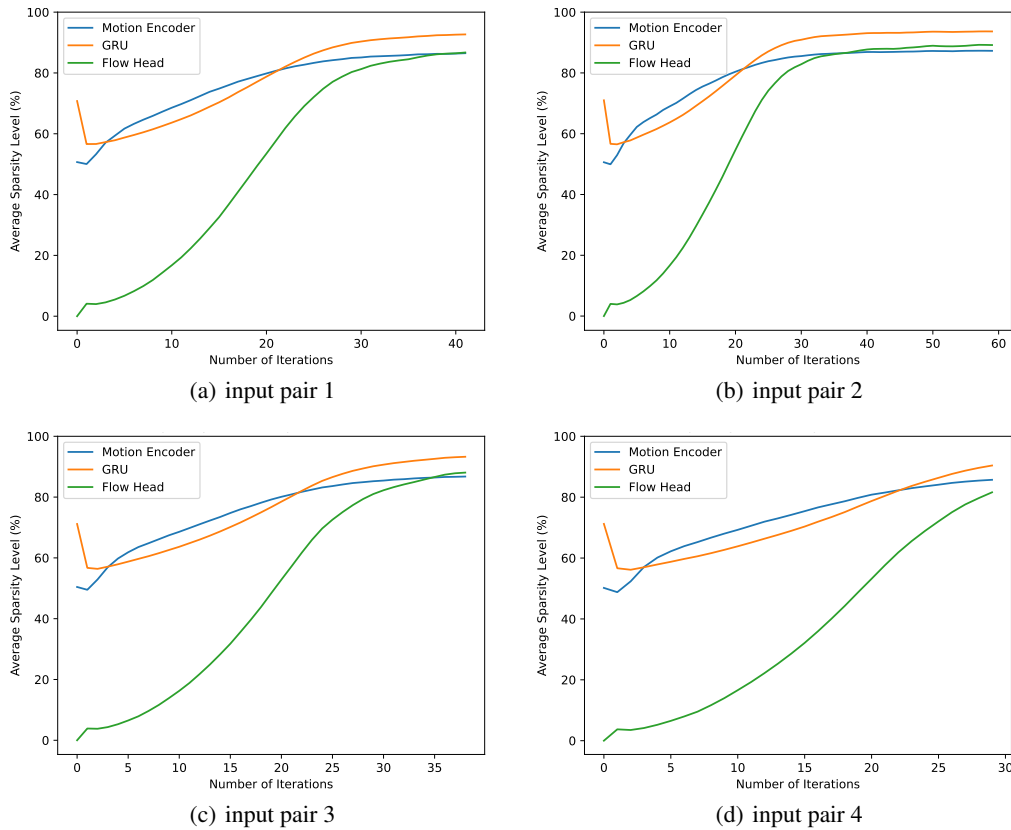

(a) input pair 1

(b) input pair 2

(c) input pair 3

(d) input pair 4

Figure 8: Averaged sparsity level of the delta feature map in different components of the update block in Fig. 10, obtained on 4 exemplary pairs of inputs. The flow head has substantially lower delta sparsity than the other two components and they all reached a very high level before the iteration converged globally.

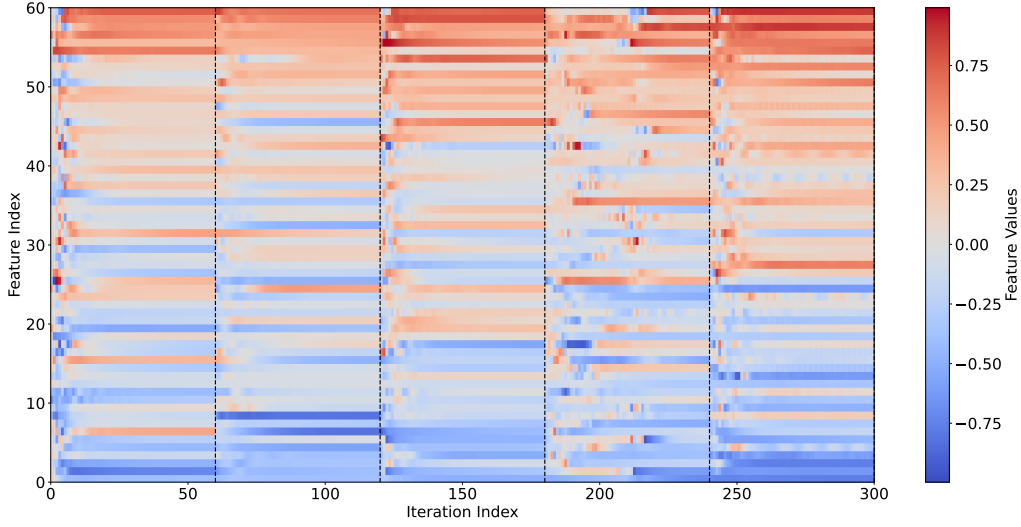

(a) with fixed-point reuse but no flow reuse

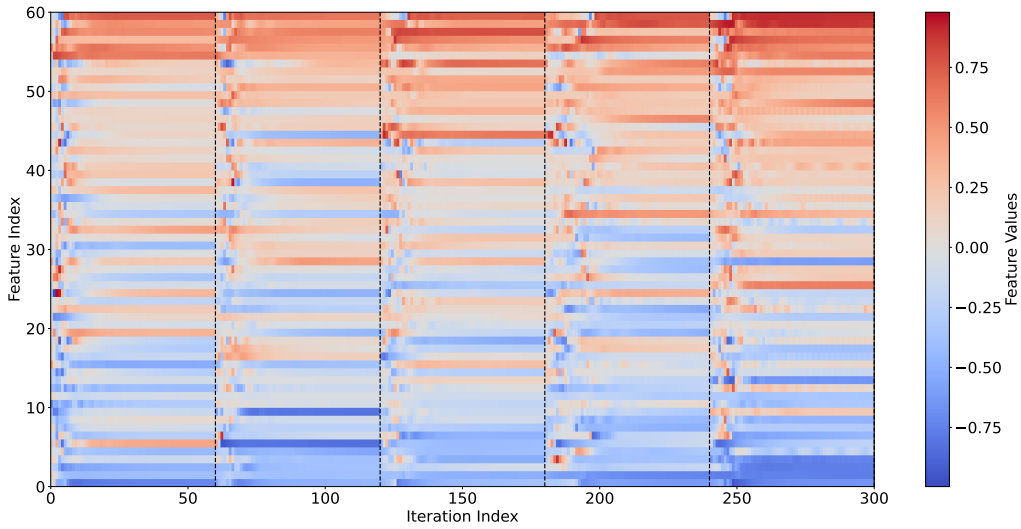

(b) with fixed-point reuse and flow reuse

Figure 9: Illustration of the evolution of activation values in the original DEQ-flow network [4] along the fixed-point iterations when processing 5 consecutive pairs of input frames in the Sintel [8] dataset. The startings of a new pair are marked with vertical dashed lines. Each pair is processed with 60 iterations. Different activation sites converge at different speeds and the fourth pair takes more iterations to reach convergence, due to the large motion in the input space.

### A.6.2 DeltaDEQ conversion details for the RAFT architecture

The original RAFT architecture [53], also as adopted in the DEQ optical flow work [4], contains mainly two parts: Fig. 10(a) the feature encoder and the context encoder. Fig. 10(b) the update block. In both RAFT and DEQ-RAFT, the update block is used to iteratively refine the flow prediction thus is the key to obtaining high-quality flow but is also the computationally expensive part. The general architecture illustration is given in Fig. 10 and our DeltaDEQ for optical flow tasks are also instantiated with this architecture. We did not apply the delta rule on the feature and context encoders since these two encoders are only inference once per input pair $x_t$ and $x_{t-1}$. But it is possible to also convert these blocks with the delta rule to further increase the computation savings. In this work, we focus on the computationally heavy part which is the update block that will be inference many times for processing every pair of inputs. The update block mainly consists of three parts as marked in orange dashed boxes: (1) motion encoder (2) flow head and (3) separable ConvGRU. The flow head outputs the delta flow $\Delta F_t^i$ which will be used to update the flow prediction $F_t^i = F_t^{i-1} + \Delta F_t^i$. We apply the delta rule to all these three components to accelerate their convolution layers with the delta rule in the DEQ fixed-point iterations.

In Fig. 8 we show the averaged sparsity level of the delta feature maps in different components of the update block when inferencing on four different input pairs. During the evolution of the fixed-point iteration, all three components' sparsity levels went up. It is worth noticing that the flow head's sparsity level started with a low level, potentially due to the fact that it is predicting a first-order difference value $\Delta F_t^i$.

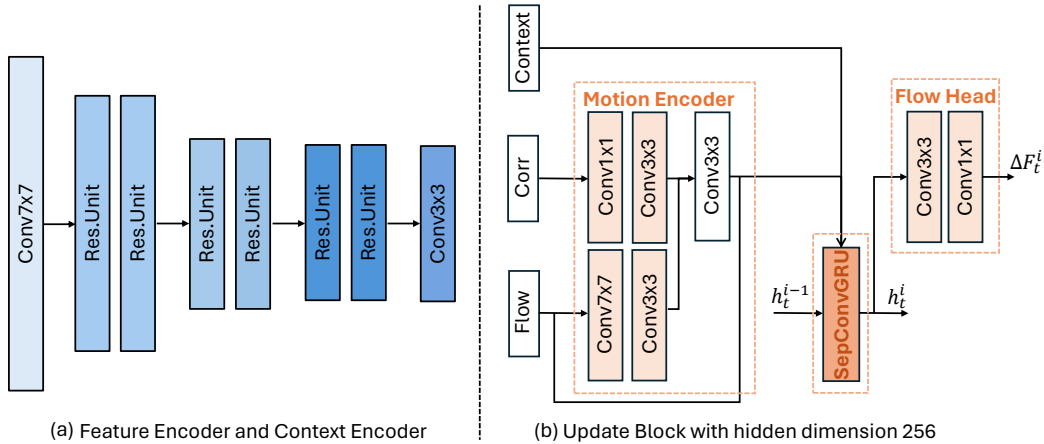

Figure 10: Architecture illustration for RAFT [53], DEQ flow [4] and our DeltaDEQ.

In our experiments, we use DEQ-RAFT-H where H represents model class huge in the work [4] with approx. 13.4 M parameters, that the hidden channels for the separable ConvGRU are 256, the output channels for the feature encoder and context encoder are both 256. We use the DEQ-RAFT-H checkpoint provided in [4] which is trained with first 120k iterations on the FlyingChairs [16] dataset and with phantom gradient [26] and 120k steps on the FlyingThings3D [40] dataset with phantom gradient. The fine-tuned DeltaDEQ-ft in Tab. 2 was further fine-tuned only on the FlyingThings3D dataset with 20k iterations and batch size 12 and learning rate 1e-6 and phantom gradient. During fine-tuning we set the weight decay to zero and did not use the FlyingChair dataset since either will degrade the accuracy. Fixed-point reuse and flow reuse are also applied.

### A.7 Comparison of fixed-point iteration and root-solving techniques in the forward pass

The DEQ series works first proposed the use of root-finding techniques to solve the fixed point in the forward pass. We argue that using fixed-point iterations including the simple Picard's iteration or the Krasnoselskii–Mann (KM) iteration suffices for obtaining a good estimation of the fixed-point $z_t^*$. Tab. 4 shows the comparison among using the solvers (Broyden, Anderson) and using fixed-point iterations (Picard, KM). We care mostly about the best possible solving quality each method could get thus we did not use early stopping nor fixed-point reuse and we increased the number of iterations until the performance increase reached a plateau. We found that fixed-point iterations actually outperformed root-solvers in terms of performance controlling the number of iterations, indicating a faster global convergence speed. Moreover, the fixed-point iterations also have an advantage in terms of the best performance reached. Thus we argue that using fixed-point iteration is a better choice for the inference of DEQ models since they not only achieve better task accuracy and have faster convergence speed, but they also require fewer computes and simpler computing patterns.

| Forward | model | iterations | Sintel (train) | | KITTI (train) | |
|---|---|---|---|---|---|---|
| | | | clean ↓ | final↓ | AEPE↓ | F1-all↓ |
| Anderson | DEQ-RAFT-H | 10 | 3.97 | 4.87 | 10.93 | 28.83 |
| | DEQ-RAFT-H | 20 | 1.90 | 3.16 | 6.02 | 17.72 |
| | DEQ-RAFT-H | 30 | 1.54 | 2.79 | 5.15 | 15.23 |
| | DEQ-RAFT-H | 40 | 1.47 | 2.72 | 4.62 | 14.11 |
| | DEQ-RAFT-H | 50 | 1.44 | 2.68 | 4.62 | 13.68 |
| | DEQ-RAFT-H | 60 | 1.43 | 2.64 | 4.37 | 13.46 |
| Broyden | DEQ-RAFT-H | 10 | 1.87 | 3.32 | 6.26 | 19.27 |
| | DEQ-RAFT-H | 20 | 1.45 | 2.70 | 4.58 | 14.26 |
| | DEQ-RAFT-H | 30 | 1.39 | 2.63 | 3.98 | 13.29 |
| | DEQ-RAFT-H | 40 | 1.36 | 2.60 | 3.85 | 13.13 |
| | DEQ-RAFT-H | 50 | 1.31 | 2.61 | 3.86 | 13.10 |
| | DEQ-RAFT-H | 60 | 1.30 | 2.59 | 3.83 | 13.07 |
| Picard | DEQ-RAFT-H | 10 | 2.03 | 3.94 | 6.02 | 18.55 |
| | DEQ-RAFT-H | 20 | 1.48 | 2.70 | 4.49 | 14.17 |
| | DEQ-RAFT-H | 30 | 1.35 | 2.63 | 3.91 | 13.22 |
| | DEQ-RAFT-H | 40 | 1.32 | 2.60 | 3.80 | 13.04 |
| | DEQ-RAFT-H | 50 | 1.32 | 2.58 | 3.78 | 12.98 |
| | DEQ-RAFT-H | 60 | 1.27 | 2.61 | 3.78 | 12.97 |
| KM $\alpha = 0.9$ | DEQ-RAFT-H | 20 | 1.42 | 2.71 | 4.56 | 14.32 |
| | DEQ-RAFT-H | 40 | 1.29 | 2.60 | 3.85 | 13.04 |
| | DEQ-RAFT-H | 60 | 1.28 | 2.59 | 3.81 | 12.97 |
| | DEQ-RAFT-H | 80 | 1.29 | 2.60 | 3.77 | 12.94 |

Table 4: Comparison among different solvers or fixed-point iteration methods for the forward pass.

## A.8 Computing platform

All experiments were conducted on an Nvidia RTX 3090 GPU with 24GB of RAM and Intel(R) Xeon(R) W-2195 CPU @ 2.30GHz.

